# Large Margin Component Analysis

**Lorenzo Torresani**
Riya, Inc.
lorenzo@riya.com

**Kuang-chih Lee**
Riya, Inc.
kclee@riya.com

## Abstract

Metric learning has been shown to significantly improve the accuracy of *k*-nearest neighbor (kNN) classification. In problems involving thousands of features, distance learning algorithms cannot be used due to overfitting and high computational complexity. In such cases, previous work has relied on a two-step solution: first apply dimensionality reduction methods to the data, and then learn a metric in the resulting low-dimensional subspace. In this paper we show that better classification performance can be achieved by unifying the objectives of dimensionality reduction and metric learning. We propose a method that solves for the low-dimensional projection of the inputs, which minimizes a metric objective aimed at separating points in different classes by a large margin. This projection is defined by a significantly smaller number of parameters than metrics learned in input space, and thus our optimization reduces the risks of overfitting. Theory and results are presented for both a linear as well as a kernelized version of the algorithm. Overall, we achieve classification rates similar, and in several cases superior, to those of support vector machines.

## 1 Introduction

The technique of *k*-nearest neighbor (kNN) is one of the most popular classification algorithms. Several reasons account for the widespread use of this method: it is straightforward to implement, it generally leads to good recognition performance thanks to the non-linearity of its decision boundaries, and its complexity is independent of the number of classes. In addition, unlike most alternatives, kNN can be applied even in scenarios where not all categories are given at the time of training, such as, for example, in face verification applications where the subjects to be recognized are not known in advance.

The distance metric defining the neighbors of a query point plays a fundamental role in the accuracy of kNN classification. In most cases Euclidean distance is used as a similarity measure. This choice is logical when it is not possible to study the statistics of the data prior to classification or when it is fair to assume that all features are equally scaled and equally relevant. However, in most cases the data is distributed in a way so that distance analysis along some specific directions of the features space can be more informative than along others. In such cases and when training data is available in advance, distance metric learning [5, 10, 4, 1, 9] has been shown to yield significant improvement in kNN classification. The key idea of these methods is to apply transformations to the data in order to emphasize the most discriminative directions. Euclidean distance computation in the transformed space is then equivalent to a non-uniform metric analysis in the original input space.

In this paper we are interested in cases where the data to be used for classification is very high-dimensional. An example is classification of imagery data, which often involves input spaces of thousands of dimensions, corresponding to the number of pixels. Metric learning in such high-dimensional spaces cannot be carried out due to overfitting and high computational complexity. In these scenarios, even kNN classification is prohibitively expensive in terms of storage and computational costs. The traditional solution is to apply dimensionality reduction methods to the data

and then learn a suitable metric in the resulting low-dimensional subspace. For example, Principal Component Analysis (PCA) can be used to compute a linear mapping that reduces the data to tractable dimensions. However, dimensionality reduction methods generally optimize objectives unrelated to classification and, as a consequence, might generate representations that are significantly less discriminative than the original data. Thus, metric learning within the subspace might lead to suboptimal similarity measures. In this paper we show that better performance can be achieved by directly solving for a low-dimensional embedding that optimizes a measure of kNN classification performance.

Our approach is inspired by the solution proposed by Weinberger et al. [9]. Their technique learns a metric that attempts to shrink distances of neighboring similarly-labeled points and to separate points in different classes by a large margin. Our contribution over previous work is twofold:

1. We describe the Large Margin Component Analysis (LMCA) algorithm, a technique that solves directly for a low-dimensional embedding of the data such that Euclidean distance in this space minimizes the large margin metric objective described in [9]. Our approach solves for only $D \cdot d$ unknowns, where $D$ is the dimensionality of the inputs and $d$ is the dimensionality of the target space. By contrast, the algorithm of Weinberger et al. [9] learns a Mahalanobis distance of the inputs, which requires solving for a $D \times D$ matrix, using iterative semidefinite programming methods. This optimization is unfeasible for large values of $D$.

2. We propose a technique that learns Mahalanobis distance metrics in nonlinear feature spaces. Our approach combines the goal of dimensionality reduction with a novel "kernelized" version of the metric learning objective of Weinberger et al. [9]. We describe an algorithm that optimizes this combined objective directly. We demonstrate that, even when data is low-dimensional and dimensionality reduction is not needed, this technique can be used to learn nonlinear metrics leading to significant improvement in kNN classification accuracy over [9].

## 2   Linear Dimensionality Reduction for Large Margin kNN Classification

In this section we briefly review the algorithm presented in [9] for metric learning in the context of kNN classification. We then describe how this approach can be generalized to compute low dimensional projections of the inputs via a novel direct optimization.

A fundamental characteristic of kNN is that its performance does not depend on linear separability of classes in input space: in order to achieve accurate kNN classification it is sufficient that the majority of the *k*-nearest points of each test example have correct label. The work of Weinberger et al. [9] exploits this property by learning a linear transformation of the input space that aims at creating consistently labeled *k*-nearest neighborhoods, i.e. clusters where each training example and its *k*-nearest points have same label and where points differently labeled are distanced by an additional safety margin. Specifically, given $n$ input examples $\mathbf{x}_1, ..., \mathbf{x}_n$ in $\Re^D$ and corresponding class labels $y_1, ..., y_n$, the technique in [9] learns the $D \times D$ transformation matrix $\mathbf{L}$ that optimizes the following objective function:

$$\epsilon(\mathbf{L}) \quad = \quad \sum_{ij} \eta_{ij} ||\mathbf{L}(\mathbf{x}_i - \mathbf{x}_j)||^2 + c \sum_{ijl} \eta_{ij}(1 - y_{il})h(||\mathbf{L}(\mathbf{x}_i - \mathbf{x}_j)||^2 - ||\mathbf{L}(\mathbf{x}_i - \mathbf{x}_l)||^2 + 1),$$

(1)

where $\eta_{ij} \in \{0, 1\}$ is a binary variable indicating whether example $\mathbf{x}_j$ is one the *k*-closest points of $\mathbf{x}_i$ that share the same label $y_i$, $c$ is a positive constant, $y_{il} \in \{0, 1\}$ is 1 iff $(y_i = y_l)$, and $h(s) = max(s, 0)$ is the hinge function. The objective $\epsilon(\mathbf{L})$ consists of two contrasting terms. The first aims at pulling closer together points sharing the same label and that were neighbors in the original space. The second term encourages distancing each example $\mathbf{x}_i$ from differently labeled points by an amount equal to 1 plus the distance from $\mathbf{x}_i$ to any of its *k* similarly-labeled closest points. This term corresponds to a margin condition similar to that of SVMs and it is used to improve generalization. The constant $c$ controls the relative importance of these two competing terms and it can be chosen via cross validation.

Upon optimization of $\epsilon(\mathbf{L})$, test example $\mathbf{x}_q$ is classified according to the kNN rule applied to its projection $\mathbf{x}_q' = \mathbf{L}\mathbf{x}_q$, using Euclidean distance as metric. Equivalently, such classification can be

interpreted as kNN classification in the original input space under the Mahalanobis distance metric induced by matrix $\mathbf{M} = \mathbf{L}^T\mathbf{L}$. Although Equation 1 is non-convex in $\mathbf{L}$, it can be rewritten as a semidefinite program $\epsilon(\mathbf{M})$ in terms of the metric $\mathbf{M}$ [9]. Thus, optimizing the objective in $\mathbf{M}$ guarantees convergence to the global minimum, regardless of initialization.

When data is very high-dimensional, minimization of $\epsilon(\mathbf{M})$ using semidefinite programming methods is impractical because of slow convergence and overfitting problems. In such cases [9] propose applying dimensionality reduction methods, such as PCA, followed by metric learning within the resulting low-dimensional subspace. As outlined above, this procedure leads to suboptimal metric learning. In this paper we propose an alternative approach that solves jointly for dimensionality reduction and metric learning. The key idea is to choose the transformation $\mathbf{L}$ in Equation 1 to be a nonsquare matrix of size $d \times D$, with $d << D$. Thus $\mathbf{L}$ defines a mapping from the high-dimensional input space to a low-dimensional embedding. Euclidean distance in this low-dimensional embedding is equivalent to Mahalanobis distance in the original input space under the rank-deficient metric $\mathbf{M} = \mathbf{L}^T\mathbf{L}$ ($\mathbf{M}$ has now rank at most $d$).

Unfortunately, optimization of $\epsilon(\mathbf{M})$ subject to rank-constraints on $\mathbf{M}$ leads to a minimization problem that is no longer convex [8] and that is awkward to solve. Here we propose an approach for minimizing the objective that differs from the one used in [9]. The idea is to optimize Equation 1 directly with respect to the nonsquare matrix $\mathbf{L}$. We argue that minimizing the objective with respect to $\mathbf{L}$ rather than with respect to the rank-deficient $D \times D$ matrix $\mathbf{M}$, offers several advantages. First, our optimization involves only $d \cdot D$ rather than $D^2$ unknowns, which considerably reduces the risk of overfitting. Second, the optimal rectangular matrix $\mathbf{L}$ computed with our method automatically satisfies the rank constraints on $\mathbf{M}$ without requiring the solution of difficult constrained minimization problems. Although the objective optimized by our method is also not convex, we experimentally demonstrate that our solution converges consistently to better metrics than those computed via the application of PCA followed by subspace distance learning (see Section 4).

We minimize $\epsilon(\mathbf{L})$ using gradient-based optimizers, such as conjugate gradient methods. Differentiating $\epsilon(\mathbf{L})$ with respect to the transformation matrix $\mathbf{L}$ gives the following gradient for the update rule:

$$
\begin{aligned}
\frac{\partial \epsilon(\mathbf{L})}{\partial \mathbf{L}} = & \; 2\mathbf{L} \sum_{ij} \eta_{ij}(\mathbf{x}_i - \mathbf{x}_j)(\mathbf{x}_i - \mathbf{x}_j)^T + \\
& \; 2c\mathbf{L} \sum_{ijl} \eta_{ij}(1 - y_{il}) \left[ (\mathbf{x}_i - \mathbf{x}_j)(\mathbf{x}_i - \mathbf{x}_j)^T - (\mathbf{x}_i - \mathbf{x}_l)(\mathbf{x}_i - \mathbf{x}_l)^T \right] \\
& \qquad h'(||\mathbf{L}(\mathbf{x}_i - \mathbf{x}_j)||^2 - ||\mathbf{L}(\mathbf{x}_i - \mathbf{x}_l)||^2 + 1)
\end{aligned}
\tag{2}
$$

We handle the non-differentiability of $h(s)$ at $s = 0$, by adopting a smooth hinge function as in [8].

## 3  Nonlinear Feature Extraction for Large Margin kNN Classification

In the previous section we have described an algorithm that jointly solves for linear dimensionality reduction and metric learning. We now describe how to "kernelize" this method in order to compute non-linear features of the inputs that optimize our distance learning objective. Our approach learns a low-rank Mahalanobis distance metric in a high dimensional feature space $F$, related to the inputs by a nonlinear map $\phi : \Re^D \to F$. We restrict our analysis to nonlinear maps $\phi$ for which there exist kernel functions $\mathbf{k}$ that can be used to compute the feature inner products without carrying out the map, i.e. such that $\mathbf{k}(\mathbf{x}_i, \mathbf{x}_j) = \phi_i^T \phi_j$, where for brevity we denoted $\phi_i = \phi(\mathbf{x}_i)$.

We modify our objective $\epsilon(\mathbf{L})$ by substituting inputs $\mathbf{x}_i$ with features $\phi(\mathbf{x}_i)$ into Equation 1. $\mathbf{L}$ is now a transformation from the space $F$ into a low-dimensional space $\Re^d$. We seek the transformation $\mathbf{L}$ minimizing the modified objective function $\epsilon(\mathbf{L})$.

The gradient in feature space can now be written as:

$$
\begin{aligned}
\frac{\partial \epsilon(\mathbf{L})}{\partial \mathbf{L}} = & \; 2 \sum_{ij} \eta_{ij} \mathbf{L}(\phi_i - \phi_j)(\phi_i - \phi_j)^T + \\
& \; 2c \sum_{ijl} \eta_{ij}(1 - y_{il}) h'(s_{ijl}) \mathbf{L} \left[ (\phi_i - \phi_j)(\phi_i - \phi_j)^T - (\phi_i - \phi_l)(\phi_i - \phi_l)^T \right]
\end{aligned}
\tag{3}
$$

where $s_{ijl} = (||\mathbf{L}(\phi_i - \phi_j)||^2 - ||\mathbf{L}(\phi_i - \phi_l)||^2 + 1)$.

Let $\Phi = [\phi_1, ..., \phi_n]^T$. We consider parameterizations of $\mathbf{L}$ of the form $\mathbf{L} = \Omega\Phi$, where $\Omega$ is some matrix allowing us to write $\mathbf{L}$ as a linear combination of the feature points. This form of nonlinear map is analogous to that used in kernel-PCA and it allows us to parameterize the transformation $\mathbf{L}$ in terms of only $d \cdot n$ parameters, the entries of the matrix $\Omega$. We now introduce the following Lemma which we will later use to derive an iterative update rule for $\mathbf{L}$.

**Lemma 3.1** *The gradient in feature space can be computed as* $\frac{\partial \epsilon(\mathbf{L})}{\partial \mathbf{L}} = \Gamma\Phi$*, where $\Gamma$ depends on features $\phi_i$ solely in terms of dot products $(\phi_i^T \phi_j)$.*

**Proof** Defining $\mathbf{k}_i = \Phi\phi_i = [\mathbf{k}(\mathbf{x}_1, \mathbf{x}_i), ..., \mathbf{k}(\mathbf{x}_n, \mathbf{x}_i)]^T$, non-linear feature projections can be computed as $\mathbf{L}\phi_i = \Omega\Phi\phi_i = \Omega\mathbf{k}_i$. From this we derive:

$$
\begin{aligned}
\frac{\partial \epsilon(\mathbf{L})}{\partial \mathbf{L}} &= 2\Omega \sum_{ij} \eta_{ij} (\mathbf{k}_i - \mathbf{k}_j)(\phi_i - \phi_j)^T + \\
&\quad 2c\Omega \sum_{ijl} \eta_{ij}(1 - y_{il})h'(s_{ijl}) \left[ (\mathbf{k}_i - \mathbf{k}_j)(\phi_i - \phi_j)^T - (\mathbf{k}_i - \mathbf{k}_l)(\phi_i - \phi_l)^T \right] \\
&= 2\Omega \sum_{ij} \eta_{ij} \left[ \mathbf{E}_i^{(\mathbf{k}_i - \mathbf{k}_j)} - \mathbf{E}_j^{(\mathbf{k}_i - \mathbf{k}_j)} \right] \Phi + \\
&\quad 2c\Omega \sum_{ijl} \eta_{ij}(1 - y_{il})h'(s_{ijl}) \left[ \mathbf{E}_i^{(\mathbf{k}_i - \mathbf{k}_j)} - \mathbf{E}_j^{(\mathbf{k}_i - \mathbf{k}_j)} - \mathbf{E}_i^{(\mathbf{k}_i - \mathbf{k}_l)} + \mathbf{E}_l^{(\mathbf{k}_i - \mathbf{k}_l)} \right] \Phi
\end{aligned}
$$

where $\mathbf{E}_i^{\mathbf{v}} = [0, ..., \mathbf{v}, 0, ..0]$ is the $n \times n$ matrix having vector $\mathbf{v}$ in the $i$-th column and all $0$ in the other columns. Setting

$$
\begin{aligned}
\Gamma &= 2\Omega \sum_{ij} \eta_{ij} \left[ \mathbf{E}_i^{(\mathbf{k}_i - \mathbf{k}_j)} - \mathbf{E}_j^{(\mathbf{k}_i - \mathbf{k}_j)} \right] + \\
&\quad 2c\Omega \sum_{ijl} \eta_{ij}(1 - y_{il})h'(s_{ijl}) \left[ \mathbf{E}_i^{(\mathbf{k}_i - \mathbf{k}_j)} - \mathbf{E}_j^{(\mathbf{k}_i - \mathbf{k}_j)} - \mathbf{E}_i^{(\mathbf{k}_i - \mathbf{k}_l)} + \mathbf{E}_l^{(\mathbf{k}_i - \mathbf{k}_l)} \right] \quad (4)
\end{aligned}
$$

proves the Lemma. ∎

This result allows us to implicitly solve for the transformation without ever computing the features in the high-dimensional space $F$: the key idea is to iteratively update $\Omega$ rather than $\mathbf{L}$. For example, using gradient descent as optimization we derive update rule:

$$
\mathbf{L}_{new} = \mathbf{L}_{old} - \lambda \left. \frac{\partial \epsilon(\mathbf{L})}{\partial \mathbf{L}} \right|_{\mathbf{L} = \mathbf{L}_{old}} = [\Omega_{old} - \lambda\Gamma_{old}] \Phi = \Omega_{new}\Phi \quad (5)
$$

where $\lambda$ is the learning rate. We carry out this optimization by iterating the update $\Omega \leftarrow (\Omega - \lambda\Gamma)$ until convergence. For classification, we project points onto the learned low-dimensional space by exploiting the kernel trick: $\mathbf{L}\phi_q = \Omega\mathbf{k}_q$.

# 4 Experimental results

We compared our methods to the metric learning algorithm of Weinberger et al. [9], which we will refer to as LMNN (Large Margin Nearest Neighbor). We use KLMCA (kernel-LMCA) to denote the nonlinear version of our algorithm. In all of the experiments reported here, LMCA was initialized using PCA, while KLMCA used the transformation computed by kernel-PCA as initial guess. The objectives of LMCA and KLMCA were optimized using the steepest descent algorithm. We experimented with more sophisticated minimization techniques, including the conjugate gradient method and the Broyden-Fletcher-Goldfarb-Shanno quasi-Newton algorithm [6], but no substantial improvement in performance or speed of convergence was achieved. The KLMCA algorithm was implemented using a Gaussian RBF kernel. The number of nearest neighbors, the weight $c$ in Equation 1, and the variance of the RBF kernel, were all automatically tuned using cross-validation.

The first part of our experimental evaluation focuses on classification results on datasets with high-dimensionality, Isolet, AT&T Faces, and StarPlus fMRI:

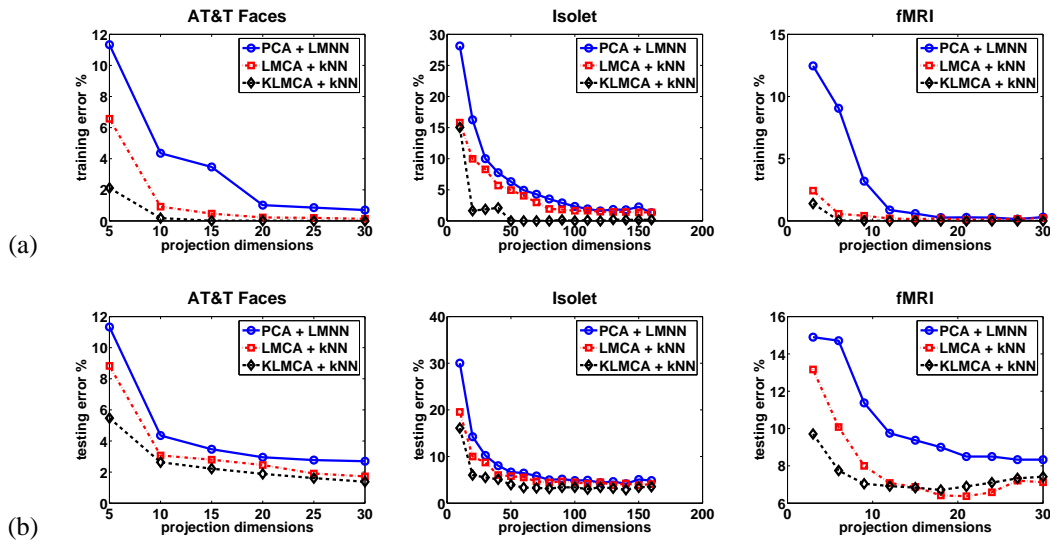

Figure 1: Classification error rates on the high-dimensional datasets Isolet, AT&T Faces and StarPlus fMRI for different projection dimensions. (a) Training error. (b) Testing error.

- Isolet[1] is a dataset of speech features from the UC Irvine repository, consisting of 6238 training examples and 1559 testing examples with 617 attributes. There are 26 classes corresponding to the spoken letters to be recognized.

- The AT&T Faces[2] database contains 10 grayscale face images of each of 40 distinct subjects. The images were taken at different times, with varying illumination, facial expressions and poses. As in [9], we downsampled the original $112 \times 92$ images to size $38 \times 31$, corresponding to 1178 input dimensions.

- The StarPlus fMRI[3] dataset contains fMRI sequences acquired in the context of a cognitive experiment. In these trials the subject is shown for a few seconds either a picture or a sentence describing a picture. The goal is to recognize the viewing activity of the subject from the fMRI images. We reduce the size of the data by considering only voxels corresponding to relevant areas of the brain cortex and by averaging the activity in each voxel over the period of the stimulus. This yields data of size 1715 for subject "04847," on which our analysis was restricted. A total number of 80 trials are available for this subject.

Except for Isolet, for which a separate testing set is specified, we computed all of the experimental results by averaging over 100 runs of random splitting of the examples into training and testing sets. For the fMRI experiment we used at each iteration 70% of the data for training and 30% for testing. For AT&T Faces, training sets were selected by sampling 7 images at random for each person. The remaining 3 images of each individual were used for testing.

Unlike LMCA and KLMCA, which directly solve for low-dimensional embeddings of the input data, LMNN cannot be run on datasets of dimensionalities such as those considered here and must be trained on lower-dimensional representations of the inputs. As in [9], we applied the LMNN algorithm on linear projections of the data computed using PCA. Figure 1 summarizes the training and testing performances of kNN classification using the metrics learned by the three algorithms for different subspace dimensions. LMCA and KLMCA give considerably better classification accuracy than LMNN on all datasets, with the kernelized version of our algorithm always outperforming the linear version. The difference in accuracy between our algorithms and LMNN is particularly dramatic when a small number of projection dimensions is used. In such cases, LMNN is unable to find good metrics in the low-dimensional subspace computed by PCA. By contrast, LMCA and KLMCA solve for the low-dimensional subspace that optimizes the classification-related objective

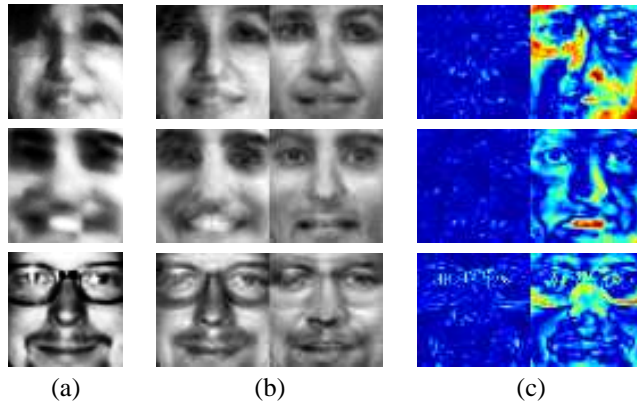

(a)                    (b)                    (c)

Figure 2: Image reconstruction from PCA and LMCA features. (a) Input images. (b) Reconstructions using PCA (left) and LMCA (right). (c) Absolute difference between original images and reconstructions from features for PCA (left) and LMCA (right). Red denotes large differences, blue indicates similar grayvalues. LMCA learns invariance to effects that are irrelevant for classification: non-uniform illumination, facial expressions, and glasses (training data contains images with and without glasses for same individuals).

of Equation 1, and therefore achieve good performance even when projecting to very low dimensions. In our experiments we found that all three classification algorithms (LMNN, LMCA+kNN, and KLMCA+kNN) performed considerably better than kNN using the Euclidean metric in the PCA and KPCA subspaces. For example, using $d = 10$ in the AT&T dataset, kNN gives a 10.9% testing error rate when used on the PCA features, and a 9.7% testing error rate when applied to the nonlinear features computed by KPCA.

While LMNN is applied to features in a low-dimensional space, LMCA and KLMCA learn a low-rank metric directly from the high-dimensional inputs. Consequently the computational complexity of our algorithms is higher than that of LMNN. However, we have found that LMCA and KLMCA converge to a minimum quite rapidly, typically within 20 iterations, and thus the complexity of these algorithms has not been a limiting factor even when applied to very high-dimensional datasets. As a reference, using $d = 10$ and $K = 3$ on the AT&T dataset, LMNN learns a metric in about 5 seconds, while LMCA and KLMCA converge to a minimum in 21 and 24 seconds, respectively.

It is instructive to look at the preimages of LMCA data embeddings. Figure 2 shows comparative reconstructions of images obtained from PCA and LMCA features by inverting their linear mappings. The PCA and LMCA subspaces in this experiment were computed from cropped face images of size $50 \times 50$ pixels, taken from a set of consumer photographs. The dataset contains 2459 face images corresponding to 152 distinct individuals. A total of $d = 125$ components were used. The subjects shown in Figure 2 were not included in the training set. For a given target dimensionality, PCA has the property of computing the linear transformation minimizing the reconstruction error under the L2 norm. Unsurprisingly, the PCA face reconstructions are extremely faithful reproductions of the original images. However, PCA accurately reconstructs also visual effects, such as lighting variations and changes in facial expressions, that are unimportant for the task of face verification and that might potentially hamper recognition. By contrast, LMCA seeks a subspace where neighboring examples belong to the same class and points differently labeled are separated by a large margin. As a result, LMCA does not encode effects that are found to be insignificant for classification or that vary largely among examples of the same class. For the case of face verification, LMCA de-emphasizes changes in illumination, presence or absence of glasses and smiling expressions (Figure 2).

When the input data does not require dimensionality reduction, LMNN and LMCA solve the same optimization problem, but LMNN should be preferred over LMCA in light of its guarantees of convergence to the global minimum of the objective. However, even in such cases, KLMCA can be used in lieu of LMNN in order to extract nonlinear features from the inputs. We have evaluated this use of KLMCA on the following low-dimensional datasets from the UCI repository: Bal, Wine, Iris, and Ionosphere. All of these datasets, except Ionosphere, have been previously used in [9] to assess the performance of LMNN. The dimensionality of the data in these sets ranges from 4 to 34. In order

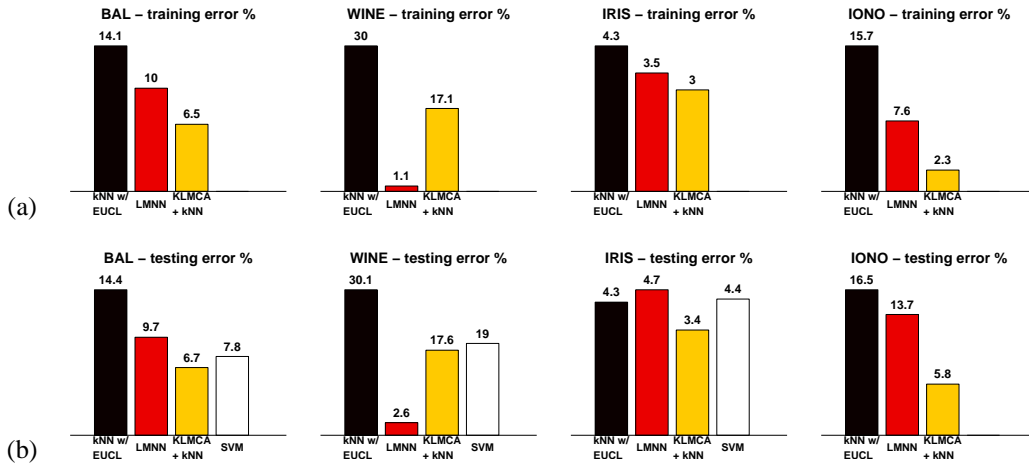

Figure 3: kNN classification accuracy on low-dimensional datasets: Bal, Wine, Iris, and Ionosphere. (a) Training error. (b) Testing error. Algorithms are kNN using Euclidean distance, LMNN [9], kNN in the nonlinear feature space computed by our KLMCA algorithm, and multiclass SVM.

to compare LMNN with KLMCA under identical conditions, KLMCA was restricted to compute a number of features equal to the input dimensionality, although in our experience using additional nonlinear features often results in better classification performance. Figure 3 summarizes the results of this comparison. Again, we averaged the errors over 100 runs with different 70/30 splits of the data for training and testing. On all datasets except on Wine, for which the mapping to the high-dimensional space seems to hurt performance (note also the high error rate of SVM), KLMCA gives better classification accuracy than LMNN. Note also that the error rates of KLMCA are consistently lower than those reported in [9] for SVM under identical training and testing conditions.

## 5   Relationship to other methods

Our method is most similar to the work of Weinberger et al. [9]. Our approach is different in focus as it specifically addresses the problem of kNN classification of very high-dimensional data. The novelty of our method lies in an optimization that solves for data reduction and metric learning simultaneously. Additionally, while [9] is limited to learning a global linear transformation of the inputs, we describe a kernelized version of our method that extracts non-linear features of the inputs. We demonstrate that this representation leads to significant improvements in kNN classification both on high-dimensional as well as on low-dimensional data. Our approach bears similarities with Linear Discriminant Analysis (LDA) [2], as both techniques solve for a low-rank Mahalanobis distance metric. However, LDA relies on the assumption that the class distributions are Gaussian and have identical covariance. These conditions are almost always violated in practice. Like our method, the Neighborhood Component Analysis (NCA) algorithm by Goldberger et al. [4] learns a low-dimensional embedding of the data for kNN classification using a direct gradient-based approach. NCA and our method differ in the definition of the objective function. Moreover, unlike our method, NCA provides purely linear embeddings of the data. A contrastive loss function analogous to the one used in this paper is adopted in [1] for training a similarity metric. A siamese architecture consisting of identical convolutional networks is used to parameterize and train the metric. In our work the metric is parameterized by arbitrary nonlinear maps for which kernel functions exist. Recent work by Globerson and Roweis [3] also proposes a technique for learning low-rank Mahalanobis metrics. Their method includes an extension for computing low-dimensional non-linear features using the kernel trick. However, this approach computes dimensionality reductions through a two-step solution which involves first solving for a possibly full-rank metric and then estimating the low-rank approximation via spectral decomposition. Besides being suboptimal, this approach is impractical for classification problems with high-dimensional data, as it requires solving for a number of unknowns that is quadratic in the number of input dimensions. Furthermore, the metric is trained with the aim of collapsing all examples in the same class to a single point. This task is difficult to achieve and not strictly necessary for good kNN classification performance. The Support Vector Decompo-

sition Machine (SVDM) [7] is also similar in spirit to our approach. SVDM optimizes an objective that is a combination of dimensionality reduction and classification. Specifically, a linear mapping from input to feature space and a linear classifier applied to feature space, are trained simultaneously. As in our work, results in their paper demonstrate that this joint optimization yields better accuracy than that achieved by learning a low-dimensional representation and a classifier separately. Unlike our method, which can be applied without any modification to classification problems with more than two classes, SVDM is formulated for binary classification only.

## 6 Discussion

We have presented a novel algorithm that simultaneously optimizes the objectives of dimensionality reduction and metric learning. Our algorithm seeks, among all possible low-dimensional projections, the one that best satisfies a large margin metric objective. Our approach contrasts techniques that are unable to learn metrics in high-dimensions and that must rely on dimensionality reduction methods to be first applied to the data. Although our optimization is not convex, we have experimentally demonstrated that the metrics learned by our solution are consistently superior to those computed by globally-optimal methods forced to search in a low-dimensional subspace.

The nonlinear version of our technique requires us to compute the kernel distance of a query point to all training examples. Future research will focus on rendering this algorithm "sparse". In addition, we will investigate methods to further reduce overfitting when learning dimensionality reduction from very high dimensions.

### Acknowledgments

We are grateful to Drago Anguelov and Burak Gokturk for discussion. We thank Aaron Hertzmann and the anonymous reviewers for their comments.

## Footnotes

[1]Available at http://www.ics.uci.edu/~mlearn/MLRepository.html

[2]Available at http://www.cl.cam.ac.uk/Research/DTG/attarchive/facedatabase.html

[3]Available at http://www.cs.cmu.edu/afs/cs.cmu.edu/project/theo-81/www/

## References

[1] S. Chopra, R. Hadsell, and Y. LeCun. Learning a similarity metric discriminatively, with application to face verification. In *Proc. IEEE Conference on Computer Vision and Pattern Recognition*, 2005.

[2] R. A. Fisher. The use of multiple measurements in taxonomic problems. *Ann. Eugenics*, 7:179–188, 1936.

[3] A. Globerson and S. Roweis. Metric learning by collapsing classes. In Y. Weiss, B. Schölkopf, and J. Platt, editors, *Advances in Neural Information Processing Systems 18*. MIT Press, Cambridge, MA, 2006.

[4] J. Goldberger, S. Roweis, G. Hinton, and R. Salakhutdinov. Neighbourhood components analysis. In L. K. Saul, Y. Weiss, and L. Bottou, editors, *Advances in Neural Information Processing Systems 17*, 2005.

[5] T. Hastie and R. Tibshirani. Discriminant adaptive nearest neighbor classification. *IEEE Transactions on Pattern Analysis and Machine Intelligence (PAMI)*, 18:607–616, 1996.

[6] A. Mordecai. *Nonlinear Programming: Analysis and Methods*. Dover Publishing, 2003.

[7] F. Pereira and G. Gordon. The support vector decomposition machine. In *Proceedings of the International Conference on Machine Learning (ICML)*, 2006.

[8] J. D. M. Rennie and N. Srebro. Fast maximum margin matrix factorization for collaborative prediction. In *Proceedings of the 22nd International Conference on Machine Learning (ICML)*, 2005.

[9] K. Q. Weinberger, J. Blitzer, and L. K. Saul. Distance metric learning for large margin nearest neighbor classification. In Y. Weiss, B. Schölkopf, and J. Platt, editors, *Advances in Neural Information Processing Systems 18*, 2006.

[10] E. P. Xing, A. Y. Ng, M. I. Jordan, , and S. Russell. Distance metric learning, with application to clustering with side-information. In T. G. Dietterich, S. Becker, and Z. Ghahramani, editors, *Advances in Neural Information Processing Systems 14*, 2002.
